# On the Computational Power of Noisy Spiking Neurons

Wolfgang Maass

Institute for Theoretical Computer Science, Technische Universitaet Graz
Klosterwiesgasse 32/2, A-8010 Graz, Austria, e-mail: maass@igi.tu-graz.ac.at

## Abstract

It has remained unknown whether one can in principle carry out reliable digital computations with networks of biologically realistic models for neurons. This article presents rigorous constructions for simulating in real-time arbitrary given boolean circuits and finite automata with arbitrarily high reliability by networks of noisy spiking neurons.

In addition we show that with the help of "shunting inhibition" even networks of very unreliable spiking neurons can simulate in real-time any McCulloch-Pitts neuron (or "threshold gate"), and therefore any multilayer perceptron (or "threshold circuit") in a reliable manner. These constructions provide a possible explanation for the fact that biological neural systems can carry out quite complex computations within 100 msec.

It turns out that the assumption that these constructions require about the shape of the EPSP's and the behaviour of the noise are surprisingly weak.

## 1  Introduction

We consider networks that consist of a finite set $V$ of *neurons*, a set $E \subseteq V \times V$ of *synapses*, a *weight* $w_{u,v} \geq 0$ and a *response function* $\varepsilon_{u,v} : \mathbf{R}^+ \to \mathbf{R}$ for each synapse

$\langle u, v \rangle \in E$ (where $\mathbf{R}^+ := \{x \in \mathbf{R} : x \geq 0\}$), and a *threshold function* $\Theta_v : \mathbf{R}^+ \to \mathbf{R}^+$ for each neuron $v \in V$.

If $F_u \subseteq \mathbf{R}^+$ is the set of *firing times* of a neuron $u$, then the *potential* at the trigger zone of neuron $v$ at time $t$ is given by $P_v(t) := \sum\limits_{u\,:\,\langle u,v\rangle \in E} \sum\limits_{s \in F_u\,:\,s < t} w_{u,v} \cdot$ $\varepsilon_{u,v}(t - s)$. The threshold function $\Theta_v(t - t')$ quantifies the "reluctance" of $v$ to fire again at time $t$, if its last previous firing was at time $t'$. We assume that $\Theta_v(0) \in (0, \infty)$, $\Theta_v(x) = \infty$ for $x \in (0, \tau_{ref}]$ (for some constant $\tau_{ref} > 0$, the "absolute refractory period"), and $\sup\{\Theta_v(x) : x \geq \tau\} < \infty$ for any $\tau > \tau_{ref}$.

In a *deterministic* model for a spiking neuron (Maass, 1995a, 1996) one can assume that a neuron $v$ fires exactly at those time points $t$ when $P_v(t)$ reaches (from below) the value $\Theta_v(t - t')$. We consider in this article a biologically more realistic model, where as in (Gerstner, van Hemmen, 1994) the size of the difference $P_v(t) - \Theta_v(t - t')$ just governs the *probability* that neuron $v$ fires. The choice of the exact firing times is left up to some unknown stochastic processes, and it may for example occur that $v$ does *not* fire in a time interval $I$ during which $P_v(t) - \Theta_v(t - t') > 0$, or that $v$ fires "spontaneously" at a time $t$ when $P_v(t) - \Theta_v(t - t') < 0$. We assume that (apart from their communication via potential changes) the stochastic processes for different neurons $v$ are independent. It turns out that the assumptions that one has to make about this stochastic firing mechanism in order to prove our results are surprisingly weak. We assume that there exist two arbitrary functions $L, \mathcal{U} : \mathbf{R} \times \mathbf{R}^+ \to [0, 1]$ so that $L(\Delta, \ell)$ provides a *lower* bound (and $\mathcal{U}(\Delta, \ell)$ provides an *upper* bound) for the probability that neuron $v$ fires during a time interval $I$ of length $\ell$ with the property that $P_v(t) - \Theta_v(t - t') \geq \Delta$ (respectively $P_v(t) - \Theta_v(t - t') \leq \Delta$) for all $t \in I$ up to the next firing of $v$ ($t'$ denotes the last firing time of $v$ *before* $I$). We just assume about these functions $L$ and $\mathcal{U}$ that they are non-decreasing in each of their two arguments (for any fixed value of the other argument), that $\lim\limits_{\Delta \to -\infty} \mathcal{U}(\Delta, \ell) = 0$ for any fixed $\ell > 0$, and that $\lim\limits_{\Delta \to \infty} L(\Delta, \ell) > 0$ for any fixed $\ell \geq R/6$ (where $R$ is the assumed length of the rising segment of an EPSP, see below). The neurons are allowed to be *"arbitrarily noisy"* in the sense that the difference $\lim\limits_{\Delta \to \infty} L(\Delta, \ell) - \lim\limits_{\Delta \to -\infty} \mathcal{U}(\Delta, \ell)$ can be arbitrarily small. Hence our constructions also apply to neurons that exhibit persistent firing failures, and they also allow for synapses that fail with a rather high probability. Furthermore a detailed analysis of our constructions shows that we can relax the somewhat dubious assumption that the noise-distributions for different neurons are independent. Thus we are also able to deal with "systematic noise" in the distribution of firing times of neurons in a pool (e.g. caused by changes in the biochemical environment that simultaneously affect *many* neurons in a pool).

It turns out that it suffices to assume only the following rather weak properties of the other functions involved in our model:

1) Each response function $\varepsilon_{u,v} : \mathbf{R}^+ \to \mathbf{R}$ is either excitatory or inhibitory (and for the sake of biological realism one may assume that each neuron $u$ induces only one type of response). All excitatory response functions $\varepsilon_{u,v}(x)$ have the value

0 for $x \in [0, \Delta_{u,v})$, and the value $\varepsilon^E(x - \Delta_{u,v})$ for $x \geq \Delta_{u,v}$, where $\Delta_{u,v} \geq 0$ is the *delay* for this synapse between neurons $u$ and $v$, and $\varepsilon^E$ is the common shape of all excitatory response functions ("EPSP's"). Corresponding assumptions are made about the inhibitory response functions ("IPSP's"), whose common shape is described by some function $\varepsilon^I : \mathbf{R}^+ \to \{x \in \mathbf{R} : x \leq 0\}$.

**2)** $\varepsilon^E$ is continuous, $\varepsilon^E(0) = 0$, $\varepsilon^E(x) = 0$ for all sufficiently large $x$, and there exists some parameter $R > 0$ such that $\varepsilon^E$ is non-decreasing in $[0, R]$, and some parameter $\rho > 0$ such that $\varepsilon^E(x + R/6) \geq \rho + \varepsilon^E(x)$ for all $x \in [0, 2R/3]$.

**3)** $-\varepsilon^I$ satisfies the same conditions as $\varepsilon^E$.

**4)** There exists a source $BN^-$ of negative *"background noise"*, that contributes to the potential $P_v(t)$ of each neuron $v$ an additive term that deviates for an arbitrarily long time interval by an arbitrarily small percentage from its average value $w_v^- \leq 0$ (which we can choose). One can delete this assumption if one assumes that the firing threshold of neurons can be shifted by some other mechanism.

In section 3 we will assume in addition the availability of a corresponding *positive* background noise $BN^+$ with average value $w_v^+ \geq 0$.
In a biological neuron $v$ one can interpret $BN^-$ and $BN^+$ as the combined effect of a continuous bombardment with a very large number of IPSP's (EPSP's) from randomly firing neurons that arrive at remote synapses on the dendritic tree of $v$.

We assume that *we* can choose the values of delays $\Delta_{u,v}$ and weights $w_{u,v}, w_v^+, w_v^-$. We refer to all assumptions specified in this section as our *"weak assumptions"* about noisy spiking neurons. It is easy to see that the most frequently studied *concrete* model for noisy spiking neurons, the *spike response model* (Gerstner and van Hemmen, 1994) satisfies these weak assumptions, and is hence a special case. However not even for the more concrete spike response model (or any other model for noisy spiking neurons) there exist any rigorous results about *computations* in these models. In fact, one may view this article as being the first that provides results about the computational complexity of neural networks for a neuron model that is acceptable to many neurobiologistis as being reasonably realistic.

In this article we only address the problem of reliable *digital* computing with noisy spiking neurons. For details of the proofs we refer to the forthcoming journal-version of this extended abstract. For results about *analog* computations with noisy spiking neurons we refer to Maass, 1995b.

## 2    Simulation of Boolean Circuits and Finite Automata with Noisy Spiking Neurons

**Theorem 1:** *For any deterministic finite automaton $D$ one can construct a network $N(D)$ consisting of any type of noisy spiking neurons that satisfy our weak assumptions, so that $N(D)$ can simulate computations of $D$ of any given length with arbitrarily high probability of correctness.*

*Idea of the* **proof:** Since the behaviour of a *single* noisy spiking neuron is completely unreliable, we use instead *pools* $A, B, \ldots$ of neurons as the basic building blocks in our construction, where all neurons $v$ in the same pool receive approximately the same "input potential" $P_v(t)$. The intricacies of our stochastic neuron model allow us only to employ a *"weak coding"* of bits, where a "1" is represented by a pool $A$ during a time interval $I$, if *at least* $p_1 \cdot |A|$ neurons in $A$ fire (at least once) during $I$ (where $p_1 > 0$ is a suitable constant), and "0" is represented if *at most* $p_0 \cdot |A|$ firings of neurons occur in $A$ during $I$, where $p_0$ with $0 < p_0 < p_1$ is another constant (that can be chosen arbitrarily small in our construction).

The described coding scheme is *weak* since it provides no useful *upper* bound (e.g. $1.5 \cdot p_1 \cdot |A|$) on the number of neurons that fire during $I$ if $A$ represents a "1" (nor on the number of firings of a single neuron in $A$). It also does not impose constraints on the *exact timing* of firings in $A$ *within* $I$. However a "0" can be represented more precisely in our model, by choosing $p_0$ sufficiently small.

The proof of Theorem 1 shows that this weak coding of bits suffices for reliable digital computations. The idea of these simulations is to introduce artificial negations into the computation, which allow us to exploit that "0" has a more precise representation than "1". It is apparently impossible to simulate an AND-gate in a straightforward fashion for a weak coding of bits, but one can simulate a NOR-gate in a reliable manner. ∎

**Corollary 2:** *Any boolean function can be computed by a sufficiently large network of noisy spiking neurons (that satisfy our weak assumptions) with arbitrarily high probability of correctness.*

# 3   Fast Simulation of Threshold Circuits via Shunting Inhibition

For biologically realistic parameters, each computation step in the previously constructed network takes around 25 msec (see point *b*) in section 4). However it is well-known that biological neural systems can carry out complex computations within just 100 msec (Churchland, Sejnowski, 1992). A closer inspection of the preceding construction shows, that one can simulate with the same speed also OR- and NOR-gates with a much larger fan-in than just 2. However wellknown results from theoretical computer science (see the results about the complexity class $AC^O$ in the survey article by Johnson in (van Leeuwen, 1990)) imply that for *any fixed* number of layers the computational power of circuits with gates for OR, NOR, AND, NOT remains very weak, even if one allows *any* polynomial size fan-in for such gates.

In contrast to that, the construction in this section will show that by using a biologically more realistic model for a noisy spiking neuron, one can in principle simulate within 100 msec 3 or more layers of a boolean circuit that employs substantially more powerful boolean gates: *threshold gates* (i.e. "Mc Culloch-Pitts neurons", also called "perceptrons"). The use of these gates provides a giant leap in computational

power for boolean circuits with a small number of layers: In spite of many years of intensive research, one has not been able to exhibit a *single concrete computational problem in the complexity classes P or NP* that can be shown to be *not* computable by a polynomial size threshold circuit with 3 layers (for threshold circuits with integer weights of *unbounded* size the same holds already for just 2 layers).

In the neuron model that we have employed so far in this article, we have assumed (as it is common in the spike response model) that the potential $P_v(t)$ at the trigger zone of neuron $v$ depends *linearly* on all the terms $w_{u,v} \cdot \varepsilon_{u,v}(t - s)$. There exists however ample biological evidence that this assumption is not appropriate for certain types of synapses. An example are synapses that carry out *shunting inhibition* (see. e.g. (Abeles, 1991) and (Shepherd, 1990)). When a synapse of this type (located on the dendritic tree of a neuron $v$) is activated, it basically erases (through a short circuit mechanism) for a short time all EPSP's that pass the location of this synapse on their way to the trigger zone of $v$. However in contrast to those IPSP's that occur linearly in the formula for $P_v(t)$, the activation of such synapse for *shunting* inhibition has *no impact* on those EPSP's that travel to the trigger zone of $v$ through another part of its dendritic tree. We model shunting inhibition in our framework as follows. We write $\Gamma$ for the subset of all neurons $\gamma$ in $V$ that can "veto" other synapses $\langle u, v \rangle$ via shunting inhibition (we assume that the neurons in $\Gamma$ have no other role apart from that). We allow in our formal model that certain $\gamma$ in $\Gamma$ are assigned as *label* to certain synapses $\langle u, v \rangle$ that have an *excitatory* response function $\varepsilon_{u,v}$. If $\gamma$ is a label of $\langle u, v \rangle$, then this models the situation that $\gamma$ can intercept EPSP's from $u$ on their way to the trigger zone of $v$ via shunting inhibition. We then define

$$P_v(t) = \sum_{u \in V \,:\, \langle u,v \rangle \in E} \left( \sum_{s \in F_u \,:\, s < t} w_{u,v} \cdot \varepsilon_{u,v}(t - s) \cdot \prod_{\gamma \text{ is label of } \langle u,v \rangle} S_\gamma(t) \right),$$

where we assume that $S_\gamma(t) \in [0, 1]$ is arbitrarily close to 0 for a short time interval after neuron $\gamma$ has fired, and else equal to 1. The firing mechanism for neurons $\gamma \in \Gamma$ is defined like for all other neurons.

**Theorem 3:** *One can simulate any threshold circuit $T$ by a sufficiently large network $N(T)$ of noisy spiking neurons with shunting inhibition (with arbitrarily high probability of correctness). The computation time of $N(T)$ does not depend on the number of gates in each layer, and is proportional to the number of layers in the threshold circuit $T$.*

*Idea of the* **proof** *of Theorem 3:* It is already impossible to simulate in a straightforward manner an AND-gate with *weak coding* of bits. The same difficulties arise in an even more drastic way if one wants to simulate a threshold gate with large fan-in.

The left part of Figure 1 indicates that with the help of shunting inhibition one can transform via an intermediate pool of neurons $B_1$ the bit that is weakly encoded by

$A_1$ into a contribution to $P_v(t)$ for neurons $v \in C$ that is throughout a time interval $J$ arbitrarily close to 0 if $A_1$ encodes a "0", and *arbitrarily close to some constant* $P^* > 0$ if $A_1$ encodes a "1" (we will call this a "*strong coding*" of a bit). Obviously it is rather easy to realize a threshold gate if one can make use of such *strong coding* of bits.

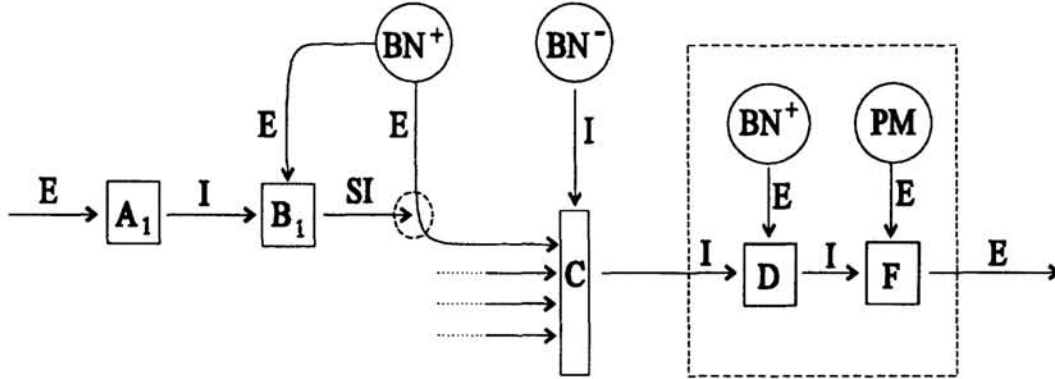

Figure 1: *Realization of a threshold gate G via shunting inhibition (SI).*

The task of the module in Figure 1 is to simulate with noisy spiking neurons a given boolean threshold gate $G$ that outputs 1 if $\sum_{i=1}^{n} \alpha_i x_i \geq \Theta$, and 0 else. For simplicity Figure 1 shows only the pool $A_1$ whose firing activity encodes (in weak coding) the first input bit $x_1$. The other input bits are represented (in weak coding) simultaneously in pools $A_2, \ldots, A_n$ parallel to $A_1$. If $x_1 = 0$, then the firing activity in pool $A_1$ is low, hence the shunting inhibition from pool $B_1$ intercepts those EPSP's that are sent from $BN^+$ to each neuron $v$ in pool $C$. More precisely, we assume that each pool $B_i$ associated with a different input bit $x_i$ carries out shunting inhibition on a *different* subtree of the dendritic tree of such neuron $v$ (where each such subtree receives EPSP's from $BN^+$). If $x_1 = 1$, the higher firing activity in pool $A_1$ inhibits the neurons in $B_1$ for some time period. Hence during the relevant time interval $BN^+$ contributes an almost constant positive summand to the potential $P_v(t)$ of neurons $v$ in $C$. By choosing $w_v^+$ and $w_v^-$ appropriately, one can achieve that during this time interval the potential $P_v(t)$ of neurons $v$ in $C$ is arbitrarily much positive if $\sum_{i=1}^{n} \alpha_i x_i \geq \Theta$, and arbitrarily much negative if $\sum_{i=1}^{n} \alpha_i x_i < \Theta$. Hence the activity level of $C$ encodes the output bit of the threshold gate $G$ (in *weak* coding). The purpose of the subsequent pools $D$ and $F$ is to synchronize (with the help of "double-negation") the output of this module via a pacemaker or synfire chain PM. In this way one can achieve that all input "bits" to another module that simulates a threshold gate on the next layer of circuit $T$ arrive simultaneously.                                                                     ∎

## 4  Conclusion

Our constructions throw new light on various experimental data, and on our attempts to understand neural computation and coding:

**a)** If one would record all firing times of a few arbitrarily chosen neurons in our networks during many repetitions of the same computation, one is likely to see that each run yields quite different seemingly random firing sequences, where however a few firing patterns will occur more frequently than could be explained by mere chance. This is consistent with the experimental results reported in (Abeles, 1991), and one should also note that the *synfire chains* of (Abeles, 1991) have many features in common with the here constructed networks.

**b)** If one plugs in biologically realistic values (see (Shepherd, 1990), (Churchland, Sejnowski, 1992)) for the length of transmission delays (around 5 msec) and the duration of EPSP's and IPSP's (around 15 msec for fast PSP's), then the computation time of our modules for NOR- and threshold gates comes out to be not more than 25 msec. Hence in principle a multi-layer perceptron with up to 4 layers can be simulated within 100 msec.

**c)** Our constructions provide new hypotheses about the computational roles of regular and shunting *inhibition*, that go far beyond their usually assumed roles.

**d)** We provide new hypotheses regarding the computational role of randomly firing neurons, and of EPSP's and IPSP's that arrive through synapses at distal parts of biological neurons (see the use of $BN^+$ and $BN^-$ in our constructions).

## References:

M. Abeles. (1991) Corticonics: Neural Circuits of the Cerebral Cortex. *Cambridge University Press.*

P. S. Churchland, T. J. Sejnowski. (1992) The Computational Brain. *MIT-Press.*

W. Gerstner, J. L. van Hemmen. (1994) How to describe neuronal activity: spikes, rates, or assemblies? *Advances in Neural Information Processing Systems, vol. 6, Morgan Kaufmann:* 463-470.

W. Maass. (1995a) On the computational complexity of networks of spiking neurons (extended abstract). *Advances in Neural Information Processing Systems, vol. 7 (Proceedings of NIPS '94), MIT-Press,* 183-190.

W. Maass. (1995b) An efficient implementation of sigmoidal neural nets in temporal coding with noisy spiking neurons. *IGI-Report 422 der Technischen Universität Graz,* submitted for publication.

W. Maass. (1996) Lower bounds for the computational power of networks of spiking neurons. *Neural Computation* 8:1, to appear.

G. M. Shepherd. (1990) The Synaptic Organization of the Brain. *Oxford University Press.*

J. van Leeuwen, ed. (1990) Handbook of Theoretical Computer Science, vol. A: Algorithms and Complexity. *MIT-Press.*
